# Modeling acoustic correlations by factor analysis

**Lawrence Saul and Mazin Rahim**
{lsaul,mazin}@research.att.com
AT&T Labs — Research
180 Park Ave, D-130
Florham Park, NJ 07932

## Abstract

Hidden Markov models (HMMs) for automatic speech recognition rely on high dimensional feature vectors to summarize the short-time properties of speech. Correlations between features can arise when the speech signal is non-stationary or corrupted by noise. We investigate how to model these correlations using factor analysis, a statistical method for dimensionality reduction. Factor analysis uses a small number of parameters to model the covariance structure of high dimensional data. These parameters are estimated by an Expectation-Maximization (EM) algorithm that can be embedded in the training procedures for HMMs. We evaluate the *combined* use of mixture densities and factor analysis in HMMs that recognize alphanumeric strings. Holding the total number of parameters fixed, we find that these methods, properly combined, yield better models than either method on its own.

## 1 Introduction

Hidden Markov models (HMMs) for automatic speech recognition[1] rely on high dimensional feature vectors to summarize the short-time, acoustic properties of speech. Though front-ends vary from recognizer to recognizer, the spectral information in each frame of speech is typically codified in a feature vector with thirty or more dimensions. In most systems, these vectors are conditionally modeled by mixtures of Gaussian probability density functions (PDFs). In this case, the correlations between different features are represented in two ways[2]: implicitly by the use of two or more mixture components, and explicitly by the non-diagonal elements in each covariance matrix. Naturally, these strategies for modeling correlations—implicit versus explicit—involve tradeoffs in accuracy, speed, and memory. This paper examines these tradeoffs using the statistical method of factor analysis.

The present work is motivated by the following observation. Currently, most HMM-based recognizers do not include any explicit modeling of correlations; that is to say—conditioned on the hidden states, acoustic features are modeled by mixtures of Gaussian PDFs with *diagonal* covariance matrices. The reasons for this practice are well known. The use of full covariance matrices imposes a heavy computational burden, making it difficult to achieve real-time recognition. Moreover, one rarely has enough data to (reliably) estimate full covariance matrices. Some of these disadvantages can be overcome by parameter-tying[3]—e.g., sharing the covariance matrices across different states or models. But parameter-tying has its own drawbacks: it considerably complicates the training procedure, and it requires some artistry to know which states should and should not be tied.

Unconstrained and diagonal covariance matrices clearly represent two extreme choices for the hidden Markov modeling of speech. The statistical method of factor analysis[4, 5] represents a compromise between these two extremes. The idea behind factor analysis is to map systematic variations of the data into a lower dimensional subspace. This enables one to represent, in a very compact way, the covariance matrices for high dimensional data. These matrices are expressed in terms of a small number of parameters that model the most significant correlations without incurring much overhead in time or memory. Maximum likelihood estimates of these parameters are obtained by an Expectation-Maximization (EM) algorithm that can be embedded in the training procedures for HMMs.

In this paper we investigate the use of factor analysis in continuous density HMMs. Applying factor analysis at the state and mixture component level[6, 7] results in a powerful form of dimensionality reduction, one tailored to the local properties of speech. Briefly, the organization of this paper is as follows. In section 2, we review the method of factor analysis and describe what makes it attractive for large problems in speech recognition. In section 3, we report experiments on the speaker-independent recognition of connected alpha-digits. Finally, in section 4, we present our conclusions as well as ideas for future research.

## 2   Factor analysis

Factor analysis is a linear method for dimensionality reduction of Gaussian random variables[4, 5]. Many forms of dimensionality reduction (including those implemented as neural networks) can be understood as variants of factor analysis. There are particularly close ties to methods based on principal components analysis (PCA) and the notion of tangent distance[8]. The combined use of mixture densities and factor analysis—resulting in a *non-linear* form of dimensionality reduction—was first applied by Hinton et al[6] to the modeling of handwritten digits. The EM procedure for mixtures of factor analyzers was subsequently derived by Ghahramani et al[7]. Below we describe the method of factor analysis for Gaussian random variables, then show how it can be applied to the hidden Markov modeling of speech.

### 2.1   Gaussian model

Let $\mathbf{x} \in \mathcal{R}^D$ denote a high dimensional Gaussian random variable. For simplicity, we will assume that $\mathbf{x}$ has zero mean. If the number of dimensions, $D$, is very large, it may be prohibitively expensive to estimate, store, multiply, or invert a full covariance matrix. The idea behind factor analysis is to find a subspace of much lower dimension, $f \ll D$, that captures most of the variations in $\mathbf{x}$. To this end, let $\mathbf{z} \in \mathcal{R}^f$ denote a low dimensional Gaussian random variable with zero mean and

identity covariance matrix:

$$P(\mathbf{z}) = \frac{1}{(2\pi)^{f/2}} e^{-z^2/2}. \tag{1}$$

We now imagine that the variable $\mathbf{x}$ is generated by a random process in which $\mathbf{z}$ is a latent (or hidden) variable; the elements of $\mathbf{z}$ are known as the *factors*. Let $\Lambda$ denote an arbitrary $D \times f$ matrix, and let $\Psi$ denote a diagonal, positive-definite $D \times D$ matrix. We imagine that $\mathbf{x}$ is generated by sampling $\mathbf{z}$ from eq. (1), computing the $D$-dimensional vector $\Lambda\mathbf{z}$, then adding independent Gaussian noise (with variances $\Psi_{ii}$) to each component of this vector. The matrix $\Lambda$ is known as the *factor loading* matrix. The relation between $\mathbf{x}$ and $\mathbf{z}$ is captured by the conditional distribution:

$$P(\mathbf{x}|\mathbf{z}) = \frac{|\Psi|^{-1/2}}{(2\pi)^{D/2}} e^{-\frac{1}{2}(\mathbf{x}-\Lambda\mathbf{z})^T \Psi^{-1}(\mathbf{x}-\Lambda\mathbf{z})}. \tag{2}$$

The marginal distribution for $\mathbf{x}$ is found by integrating out the hidden variable $\mathbf{z}$. The calculation is straightforward because both $P(\mathbf{z})$ and $P(\mathbf{x}|\mathbf{z})$ are Gaussian:

$$P(\mathbf{x}) = \int d\mathbf{z} \, P(\mathbf{x}|\mathbf{z})P(\mathbf{z}) \tag{3}$$

$$= \frac{|\Psi + \Lambda\Lambda^T|^{-1/2}}{(2\pi)^{D/2}} e^{-\frac{1}{2}\mathbf{x}^T(\Psi+\Lambda\Lambda^T)^{-1}\mathbf{x}}. \tag{4}$$

From eq. (4), we see that $\mathbf{x}$ is normally distributed with mean zero and covariance matrix $\Psi + \Lambda\Lambda^T$. It follows that when the diagonal elements of $\Psi$ are small, most of the variation in $\mathbf{x}$ occurs in the subspace spanned by the columns of $\Lambda$. The variances $\Psi_{ii}$ measure the typical size of componentwise fluctuations outside this subspace.

Covariance matrices of the form $\Psi + \Lambda\Lambda^T$ have a number of useful properties. Most importantly, they are expressed in terms of a small number of parameters, namely the $D(f+1)$ non-zero elements of $\Lambda$ and $\Psi$. If $f \ll D$, then storing $\Lambda$ and $\Psi$ requires much less memory than storing a full covariance matrix. Likewise, estimating $\Lambda$ and $\Psi$ also requires much less data than estimating a full covariance matrix. Covariance matrices of this form can be efficiently inverted using the matrix inversion lemma[9],

$$(\Psi + \Lambda\Lambda^T)^{-1} = \Psi^{-1} - \Psi^{-1}\Lambda(I + \Lambda^T\Psi^{-1}\Lambda)^{-1}\Lambda^T\Psi^{-1} \tag{5}$$

where $I$ is the $f \times f$ identity matrix. This decomposition also allows one to compute the probability $P(\mathbf{x})$ with only $O(fD)$ multiplies, as opposed to the $O(D^2)$ multiplies that are normally required when the covariance matrix is non-diagonal.

Maximum likelihood estimates of the parameters $\Lambda$ and $\Psi$ are obtained by an EM procedure[4]. Let $\{\mathbf{x}_t\}$ denote a sample of data points (with mean zero). The EM procedure is an iterative procedure for maximizing the log-likelihood, $\sum_t \ln P(\mathbf{x}_t)$, with $P(\mathbf{x}_t)$ given by eq. (4). The E-step of this procedure is to compute:

$$Q(\Lambda', \Psi'; \Lambda, \Psi) = \sum_t \int d\mathbf{z} \, P(\mathbf{z}|\mathbf{x}_t, \Lambda, \Psi) \ln P(\mathbf{z}, \mathbf{x}_t|\Lambda', \Psi'). \tag{6}$$

The right hand side of eq. (6) depends on $\Lambda$ and $\Psi$ through the statistics[7]:

$$\mathrm{E}[\mathbf{z}|\mathbf{x}_t] = [I + \Lambda^T\Psi^{-1}\Lambda]^{-1}\Lambda^T\Psi^{-1}\mathbf{x}_t, \tag{7}$$

$$\mathrm{E}[\mathbf{z}\mathbf{z}^T|\mathbf{x}_t] = [I + \Lambda^T\Psi^{-1}\Lambda]^{-1} + \mathrm{E}[\mathbf{z}|\mathbf{x}_t]\mathrm{E}[\mathbf{z}^T|\mathbf{x}_t]. \tag{8}$$

Here, $\mathrm{E}[\cdot|\mathbf{x}_t]$ denotes an average with respect to the posterior distribution, $P(\mathbf{z}|\mathbf{x}_t, \Lambda, \Psi)$. The M-step of the EM algorithm is to maximize the right hand

side of eq. (6) with respect to $\Psi'$ and $\Lambda'$. This leads to the iterative updates[7]:

$$\Lambda' = \left( \sum_t \mathbf{x}_t \mathrm{E}[\mathbf{z}^T | \mathbf{x}_t] \right) \left( \sum_t \mathrm{E}[\mathbf{z}\mathbf{z}^T | \mathbf{x}_t] \right)^{-1}, \tag{9}$$

$$\Psi' = \mathrm{diag} \left\{ \frac{1}{N} \sum_t \left[ \mathbf{x}_t \mathbf{x}_t^T - \Lambda' \mathrm{E}[\mathbf{z}|\mathbf{x}_t] \mathbf{x}_t^T \right] \right\}, \tag{10}$$

where $N$ is the number of data points, and $\Psi'$ is constrained to be purely diagonal. These updates are guaranteed to converge monotonically to a (possibly local) maximum of the log-likelihood.

## 2.2 Hidden Markov modeling of speech

Consider a continuous density HMM whose feature vectors, conditioned on the hidden states, are modeled by mixtures of Gaussian PDFs. If the dimensionality of the feature space is very large, we can make use of the parameterization in eq. (4). Each mixture component thus obtains its own means, variances, and factor loading matrix. Taken together, these amount to a total of $C(f + 2)D$ parameters per mixture model, where $C$ is the number of mixture components, $f$ the number of factors, and $D$ the dimensionality of the feature space. Note that these models capture feature correlations in two ways: implicitly, by using two or more mixture components, and explicitly, by using one or more factors. Intuitively, one expects the mixture components to model *discrete* types of variability (e.g., whether the speaker is male or female), and the factors to model *continuous* types of variability (e.g., due to coarticulation or noise). Both types of variability are important for building accurate models of speech.

It is straightforward to integrate the EM algorithm for factor analysis into the training of HMMs. Suppose that $\mathcal{S} = \{\mathbf{x}_t\}$ represents a sequence of acoustic vectors. The forward-backward procedure enables one to compute the posterior probability, $\gamma_t^{sc} = P(s_t = s, c_t = c | \mathcal{S})$, that the HMM used state $s$ and mixture component $c$ at time $t$. The updates for the matrices $\Lambda^{sc}$ and $\Psi^{sc}$ (within each state and mixture component) have essentially the same form as eqs. (9-10), except that now each observation $\mathbf{x}_t$ is weighted by the posterior probability, $\gamma_t^{sc}$. Additionally, one must take into account that the mixture components have non-zero means[7]. A complete derivation of these updates (along with many additional details) will be given in a longer version of this paper.

Clearly, an important consideration when applying factor analysis to speech is the choice of acoustic features. A standard choice—and the one we use in our experiments—is a thirty-nine dimensional feature vector that consists of twelve cepstral coefficients (with first and second derivatives) and the normalized log-energy (with first and second derivatives). There are known to be correlations[2] between these features, especially between the different types of coefficients (e.g., cepstrum and delta-cepstrum). While these correlations have motivated our use of factor analysis, it is worth emphasizing that the method applies to arbitrary feature vectors. Indeed, whatever features are used to summarize the short-time properties of speech, one expects correlations to arise from coarticulation, background noise, speaker idiosyncrasies, etc.

## 3 Experiments

Continuous density HMMs with diagonal and factored covariance matrices were trained to recognize alphanumeric strings (e.g., N Z 3 V J 4 E 3 U 2). Highly

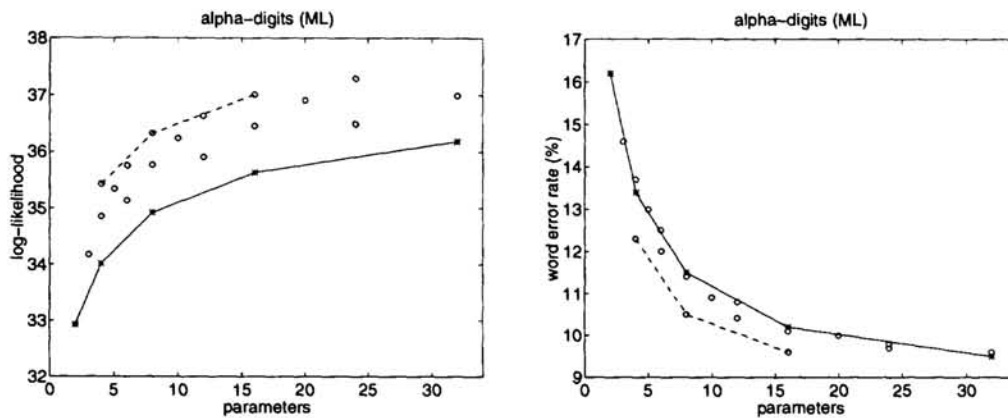

Figure 1: Plots of log-likelihood scores and word error rates on the test set versus the number of parameters per mixture model (divided by the number of features). The stars indicate models with diagonal covariance matrices; the circles indicate models with factor analysis. The dashed lines connect the recognizers in table 2.

confusable letters such as B/V, C/Z, and M/N make this a challenging problem in speech recognition. The training and test data were recorded over a telephone network and consisted of 14622 and 7255 utterances, respectively. Recognizers were built from 285 left-to-right HMMs trained by maximum likelihood estimation; each HMM modeled a context-dependent sub-word unit. Testing was done with a free grammar network (i.e., no grammar constraints). We ran several experiments, varying both the number of mixture components and the number of factors. The goal was to determine the best model of acoustic feature correlations.

Table 1 summarizes the results of these experiments. The columns from left to right show the number of mixture components, the number of factors, the number of parameters per mixture model (divided by the feature dimension), the word error rates (including insertion, deletion, and substition errors) on the test set, the average log-likelihood per frame of speech on the test set, and the CPU time to recognize twenty test utterances (on an SGI R4000). Not surprisingly, the word accuracies and likelihood scores increase with the number of modeling parameters; likewise, so do the CPU times. The most interesting comparisons are between models with the same number of parameters—e.g., four mixture components with no factors versus two mixture components with two factors. The left graph in figure 1 shows a plot of the average log-likelihood versus the number of parameters per mixture model; the stars and circles in this plot indicate models with and without diagonal covariance matrices. One sees quite clearly from this plot that given a *fixed* number of parameters, models with non-diagonal (factored) covariance matrices tend to have higher likelihoods. The right graph in figure 1 shows a similar plot of the word error rates versus the number of parameters. Here one does not see much difference; presumably, because HMMs are such poor models of speech to begin with, higher likelihoods do not necessarily translate into lower error rates. We will return to this point later.

It is worth noting that the above experiments used a *fixed* number of factors per mixture component. In fact, because the variability of speech is highly context-dependent, it makes sense to vary the number of factors, even across states within the same HMM. A simple heuristic is to adjust the number of factors depending on the amount of training data for each state (as determined by an initial segmentation of the training utterances). We found that this heuristic led to more pronounced

| $C$ | $f$ | $C(f+2)$ | word error (%) | log-likelihood | CPU time (sec) |
|---|---|---|---|---|---|
| 1 | 0 | 2 | 16.2 | 32.9 | 25 |
| 1 | 1 | 3 | 14.6 | 34.2 | 30 |
| 1 | 2 | 4 | 13.7 | 34.9 | 30 |
| 1 | 3 | 5 | 13.0 | 35.3 | 38 |
| 1 | 4 | 6 | 12.5 | 35.8 | 39 |
| 2 | 0 | 4 | 13.4 | 34.0 | 30 |
| 2 | 1 | 6 | 12.0 | 35.1 | 44 |
| 2 | 2 | 8 | 11.4 | 35.8 | 48 |
| 2 | 3 | 10 | 10.9 | 36.2 | 61 |
| 2 | 4 | 12 | 10.8 | 36.6 | 67 |
| 4 | 0 | 8 | 11.5 | 34.9 | 46 |
| 4 | 1 | 12 | 10.4 | 35.9 | 80 |
| 4 | 2 | 16 | 10.1 | 36.5 | 93 |
| 4 | 3 | 20 | 10.0 | 36.9 | 132 |
| 4 | 4 | 24 | 9.8 | 37.3 | 153 |
| 8 | 0 | 16 | 10.2 | 35.6 | 93 |
| 8 | 1 | 24 | 9.7 | 36.5 | 179 |
| 8 | 2 | 32 | 9.6 | 37.0 | 226 |
| 16 | 0 | 32 | 9.5 | 36.2 | 222 |

Table 1: Results for different recognizers. The columns indicate the number of mixture components, the number of factors, the number of parameters per mixture model (divided by the number of features), the word error rates and average log-likelihood scores on the test set, and the CPU time to recognize twenty utterances.

| $C$ | $f$ | $C(f+2)$ | word error (%) | log-likelihood | CPU time (sec) |
|---|---|---|---|---|---|
| 1 | 2 | 4 | 12.3 | 35.4 | 32 |
| 2 | 2 | 8 | 10.5 | 36.3 | 53 |
| 4 | 2 | 16 | 9.6 | 37.0 | 108 |

Table 2: Results for recognizers with variable numbers of factors; $f$ denotes the *average* number of factors per mixture component.

differences in likelihood scores and error rates. In particular, substantial improvements were observed for three recognizers whose HMMs employed an *average* of two factors per mixture component; see the dashed lines in figure 1. Table 2 summarizes these results. The reader will notice that these recognizers are extremely competitive in all aspects of performance—accuracy, memory, and speed—with the baseline (zero factor) models in table 1.

## 4    Discussion

In this paper we have studied the combined use of mixture densities and factor analysis for speech recognition. This was done in the framework of hidden Markov modeling, where acoustic features are conditionally modeled by mixtures of Gaussian PDFs. We have shown that mixture densities and factor analysis are complementary means of modeling acoustic correlations. Moreover, when used together, they can lead to smaller, faster, and more accurate recognizers than either method on its own. (Compare the last lines of tables 1 and 2.)

Several issues deserve further investigation. First, we have seen that increases in likelihood scores do not always correspond to reductions in error rates. (This is a common occurrence in automatic speech recognition.) We are currently investigating *discriminative* methods[10] for training HMMs with factor analysis; the idea here is to optimize an objective function that more directly relates to the goal of minimizing classification errors. Second, it is important to extend our results to large vocabulary tasks in speech recognition. The extreme sparseness of data in these tasks makes factor analysis an appealing strategy for dimensionality reduction. Finally, there are other questions that need to be answered. Given a limited number of parameters, what is the best way to allocate them among factors *and* mixture components? Do the cepstral features used by HMMs throw away informative correlations in the speech signal? Could such correlations be better modeled by factor analysis? Answers to these questions can only lead to further improvements in overall performance.

## Acknowledgements

We are grateful to A. Ljolje (AT&T Labs), Z. Ghahramani (University of Toronto) and H. Seung (Bell Labs) for useful discussions. We also thank P. Modi (AT&T Labs) for providing an initial segmentation of the training utterances.

# References

[1] Rabiner, L., and Juang, B. (1993) *Fundamentals of Speech Recognition.* Englewood Cliffs: Prentice Hall.

[2] Ljolje, A. (1994) The importance of cepstral parameter correlations in speech recognition. *Computer Speech and Language* **8**:223-232.

[3] Bellegarda, J., and Nahamoo, D. (1990) Tied mixture continuous parameter modeling for speech recognition. *IEEE Transactions on Acoustics, Speech, and Signal Processing* **38**:2033-2045.

[4] Rubin, D., and Thayer, D. (1982) EM algorithms for factor analysis. *Psychometrika* **47**:69-76.

[5] Everitt, B. (1984) *An introduction to latent variable models.* London: Chapman and Hall.

[6] Hinton, G., Dayan, P., and Revow, M. (1996) Modeling the manifolds of images of handwritten digits. To appear in *IEEE Transactions on Neural Networks.*

[7] Ghahramani, Z. and Hinton, G. (1996) The EM algorithm for mixtures of factor analyzers. *University of Toronto Technical Report* CRG-TR-96-1.

[8] Simard, P., LeCun, Y., and Denker, J. (1993) Efficient pattern recognition using a new transformation distance. In J. Cowan, S. Hanson, and C. Giles, eds. *Advances in Neural Information Processing Systems* **5**:50-58. Cambridge: MIT Press.

[9] Press, W., Teukolsky, S., Vetterling, W., and Flannery, B. (1992) *Numerical Recipes in C: The Art of Scientific Computing.* Cambridge: Cambridge University Press.

[10] Bahl, L., Brown, P., deSouza, P., and Mercer, L. (1986) Maximum mutual information estimation of hidden Markov model parameters for speech recognition. In *Proceedings of ICASSP 86*: 49-52.